# Error Propagation for Approximate Policy and Value Iteration

**Amir massoud Farahmand**
Department of Computing Science
University of Alberta
Edmonton, Canada, T6G 2E8
amirf@ualberta.ca

**Rémi Munos**
Sequel Project, INRIA Lille
Lille, France
remi.munos@inria.fr

**Csaba Szepesvári** *
Department of Computing Science
University of Alberta
Edmonton, Canada, T6G 2E8
szepesva@ualberta.ca

## Abstract

We address the question of how the approximation error/Bellman residual at each iteration of the Approximate Policy/Value Iteration algorithms influences the quality of the resulted policy. We quantify the performance loss as the $L_p$ norm of the approximation error/Bellman residual at each iteration. Moreover, we show that the performance loss depends on the expectation of the squared Radon-Nikodym derivative of a certain distribution rather than its supremum – as opposed to what has been suggested by the previous results. Also our results indicate that the contribution of the approximation/Bellman error to the performance loss is more prominent in the later iterations of API/AVI, and the effect of an error term in the earlier iterations decays exponentially fast.

## 1 Introduction

The exact solution for the reinforcement learning (RL) and planning problems with large state space is difficult or impossible to obtain, so one usually has to aim for approximate solutions. Approximate Policy Iteration (API) and Approximate Value Iteration (AVI) are two classes of iterative algorithms to solve RL/Planning problems with large state spaces. They try to approximately find the fixed-point solution of the Bellman optimality operator.

AVI starts from an initial value function $V_0$ (or $Q_0$), and iteratively applies an *approximation* of $T^*$, the Bellman optimality operator, (or $T^\pi$ for the policy evaluation problem) to the previous estimate, i.e., $V_{k+1} \approx T^* V_k$. In general, $V_{k+1}$ is not equal to $T^* V_k$ because (1) we do not have direct access to the Bellman operator but only some samples from it, and (2) the function space in which $V$ belongs is not representative enough. Thus there would be an approximation error $\varepsilon_k = T^* V_k - V_{k+1}$ between the result of the exact VI and AVI.

Some examples of AVI-based approaches are tree-based Fitted Q-Iteration of Ernst et al. [1], multi-layer perceptron-based Fitted Q-Iteration of Riedmiller [2], and regularized Fitted Q-Iteration of Farahmand et al. [3]. See the work of Munos and Szepesvári [4] for more information about AVI.

API is another iterative algorithm to find an approximate solution to the fixed point of the Bellman *optimality* operator. It starts from a policy $\pi_0$, and then approximately *evaluates* that policy $\pi_0$, i.e., it finds a $Q_0$ that satisfies $T^{\pi_0}Q_0 \approx Q_0$. Afterwards, it performs a policy improvement step, which is to calculate the greedy policy with respect to (w.r.t.) the most recent action-value function, to get a new policy $\pi_1$, i.e., $\pi_1(\cdot) = \arg\max_{a \in \mathcal{A}} Q_0(\cdot, a)$. The policy iteration algorithm continues by approximately evaluating the newly obtained policy $\pi_1$ to get $Q_1$ and repeating the whole process again, generating a sequence of policies and their corresponding approximate action-value functions $Q_0 \rightarrow \pi_1 \rightarrow Q_1 \rightarrow \pi_2 \rightarrow \cdots$. Same as AVI, we may encounter a difference between the approximate solution $Q_k$ ($T^{\pi_k}Q_k \approx Q_k$) and the true value of the policy $Q^{\pi_k}$, which is the solution of the fixed-point equation $T^{\pi_k}Q^{\pi_k} = Q^{\pi_k}$. Two convenient ways to describe this error is either by the Bellman residual of $Q_k$ ($\varepsilon_k = Q_k - T^{\pi_k}Q_k$) or the policy evaluation approximation error ($\varepsilon_k = Q_k - Q^{\pi_k}$).

API is a popular approach in RL literature. One well-known algorithm is LSPI of Lagoudakis and Parr [5] that combines Least-Squares Temporal Difference (LSTD) algorithm (Bradtke and Barto [6]) with a policy improvement step. Another API method is to use the Bellman Residual Minimization (BRM) and its variants for policy evaluation and iteratively apply the policy improvement step (Antos et al. [7], Maillard et al. [8]). Both LSPI and BRM have many extensions: Farahmand et al. [9] introduced a nonparametric extension of LSPI and BRM and formulated them as an optimization problem in a reproducing kernel Hilbert space and analyzed its statistical behavior. Kolter and Ng [10] formulated an $l_1$ regularization extension of LSTD. See Xu et al. [11] and Jung and Polani [12] for other examples of kernel-based extension of LSTD/LSPI, and Taylor and Parr [13] for a unified framework. Also see the proto-value function-based approach of Mahadevan and Maggioni [14] and iLSTD of Geramifard et al. [15].

A crucial question in the applicability of API/AVI, which is the main topic of this work, is to understand how either the approximation error or the Bellman residual at each iteration of API or AVI affects the quality of the resulted policy. Suppose we run API/AVI for $K$ iterations to obtain a policy $\pi_K$. Does the knowledge that all $\varepsilon_k$s are small (maybe because we have had a lot of samples and used powerful function approximators) imply that $V^{\pi_K}$ is close to the optimal value function $V^*$ too? If so, how does the errors occurred at a certain iteration $k$ **propagate** through iterations of API/AVI and affect the final performance loss?

There have already been some results that partially address this question. As an example, Proposition 6.2 of Bertsekas and Tsitsiklis [16] shows that for API applied to a finite MDP, we have $\limsup_{k \to \infty} \|V^* - V^{\pi_k}\|_\infty \leq \frac{2\gamma}{(1-\gamma)^2} \limsup_{k \to \infty} \|V^{\pi_k} - V_k\|_\infty$ where $\gamma$ is the discount facto. Similarly for AVI, if the approximation errors are uniformly bounded ($\|T^* V_k - V_{k+1}\|_\infty \leq \varepsilon$), we have $\limsup_{k \to \infty} \|V^* - V^{\pi_k}\|_\infty \leq \frac{2\gamma}{(1-\gamma)^2}\varepsilon$ (Munos [17]).

Nevertheless, most of these results are pessimistic in several ways. One reason is that they are expressed as the supremum norm of the approximation errors $\|V^{\pi_k} - V_k\|_\infty$ or the Bellman error $\|Q_k - T^{\pi_k}Q_k\|_\infty$. Compared to $L_p$ norms, the supremum norm is conservative. It is quite possible that the result of a learning algorithm has a small $L_p$ norm but a very large $L_\infty$ norm. Therefore, it is desirable to have a result expressed in $L_p$ norm of the approximation/Bellman residual $\varepsilon_k$.

In the past couple of years, there have been attempts to extend $L_\infty$ norm results to $L_p$ ones [18, 17, 7]. As a typical example, we quote the following from Antos et al. [7]:

**Proposition 1** (Error Propagation for API – [7]). *Let $p \geq 1$ be a real and $K$ be a positive integer. Then, for any sequence of functions $\{Q^{(k)}\} \subset B(\mathcal{X} \times \mathcal{A}; Q_{max})(0 \leq k < K)$, the space of $Q_{max}$-bounded measurable functions, and their corresponding Bellman residuals $\varepsilon_k = Q_k - T^\pi Q_k$, the following inequalities hold:*

$$\|Q^* - Q^{\pi_K}\|_{p,\rho} \leq \frac{2\gamma}{(1-\gamma)^2}\left(C_{\rho,\nu}^{1/p} \max_{0 \leq k < K} \|\varepsilon_k\|_{p,\nu} + \gamma^{\frac{K}{p}-1} R_{max}\right),$$

*where $R_{max}$ is an upper bound on the magnitude of the expected reward function and*

$$C_{\rho,\nu} = (1-\gamma)^2 \sum_{m \geq 1} m\gamma^{m-1} \sup_{\pi_1,\ldots,\pi_m} \left\|\frac{d\left(\rho P^{\pi_1} \cdots P^{\pi_m}\right)}{d\nu}\right\|_\infty.$$

This result indeed uses $L_p$ norm of the Bellman residuals and is an improvement over results like Bertsekas and Tsitsiklis [16, Proposition 6.2], but still is pessimistic in some ways and does

not answer several important questions. For instance, this result implies that the uniform-over-all-iterations upper bound $\max_{0 \le k < K} \|\varepsilon_k\|_{p,\nu}$ is the quantity that determines the performance loss. One may wonder if this condition is really necessary, and ask whether it is better to put more emphasis on earlier/later iterations? Or another question is whether the appearance of terms in the form of $\|\frac{d(\rho P^{\pi_1} \cdots P^{\pi_m})}{d\nu}\|_\infty$ is intrinsic to the difficulty of the problem or can be relaxed.

The goal of this work is to answer these questions and to provide tighter upper bounds on the performance loss of API/AVI algorithms. These bounds help one understand what factors contribute to the difficulty of a learning problem. We base our analysis on the work of Munos [17], Antos et al. [7], Munos [18] and provide upper bounds on the performance loss in the form of $\|V^* - V^{\pi_k}\|_{1,\rho}$ (the expected loss weighted according to the evaluation probability distribution $\rho$ – this is defined in Section 2) for API (Section 3) and AVI (Section 4). This performance loss depends on a certain function of $\nu$-weighted $L_2$ norms of $\varepsilon_k$s, in which $\nu$ is the data sampling distribution, and $C_{\rho,\nu}(K)$ that depends on the MDP, two probability distributions $\rho$ and $\nu$, and the number of iterations $K$.

In addition to relating the performance loss to $L_p$ norm of the Bellman residual/approximation error, this work has three main contributions that to our knowledge have not been considered before: (1) We show that the performance loss depends on the **expectation** of the squared Radon-Nikodym derivative of a certain distribution, to be specified in Section 3, rather than its supremum. The difference between this expectation and the supremum can be considerable. For instance, for a finite state space with $N$ states, the ratio can be of order $O(N^{1/2})$. (2) The contribution of the Bellman/approximation error to the performance loss is more prominent in later iterations of API/AVI, and the effect of an error term in early iterations decays exponentially fast. (3) There are certain structures in the definition of concentrability coefficients that have not been explored before. We thoroughly discuss these **qualitative/structural** improvements in Section 5.

## 2  Background

In this section, we provide a very brief summary of some of the concepts and definitions from the theory of Markov Decision Processes (MDP) and reinforcement learning (RL) and a few other notations. For further information about MDPs and RL the reader is referred to [19, 16, 20, 21].

A *finite-action discounted MDP* is a 5-tuple $(\mathcal{X}, \mathcal{A}, P, \mathcal{R}, \gamma)$, where $\mathcal{X}$ is a measurable state space, $\mathcal{A}$ is a finite set of actions, $P$ is the probability transition kernel, $\mathcal{R}$ is the reward kernel, and $0 \le \gamma < 1$ is the discount factor. The transition kernel $P$ is a mapping with domain $\mathcal{X} \times \mathcal{A}$ evaluated at $(x, a) \in \mathcal{X} \times \mathcal{A}$ that gives a distribution over $\mathcal{X}$, which we shall denote by $P(\cdot|x, a)$. Likewise, $\mathcal{R}$ is a mapping with domain $\mathcal{X} \times \mathcal{A}$ that gives a distribution of immediate reward over $\mathbb{R}$, which is denoted by $\mathcal{R}(\cdot|x, a)$. We denote $r(x, a) = \mathbb{E}\left[\mathcal{R}(\cdot|x, a)\right]$, and assume that its absolute value is bounded by $R_{\max}$.

A mapping $\pi : \mathcal{X} \to \mathcal{A}$ is called a deterministic Markov stationary policy, or just a *policy* in short. Following a policy $\pi$ in an MDP means that at each time step $A_t = \pi(X_t)$. Upon taking action $A_t$ at $X_t$, we receive reward $R_t \sim \mathcal{R}(\cdot|x, a)$, and the Markov chain evolves according to $X_{t+1} \sim P(\cdot|X_t, A_t)$. We denote the probability transition kernel of following a policy $\pi$ by $P^\pi$, i.e., $P^\pi(dy|x) = P(dy|x, \pi(x))$.

The value function $V^\pi$ for a policy $\pi$ is defined as $V^\pi(x) \triangleq \mathbb{E}\left[\sum_{t=0}^\infty \gamma^t R_t \,\middle|\, X_0 = x\right]$ and the action-value function is defined as $Q^\pi(x, a) \triangleq \mathbb{E}\left[\sum_{t=0}^\infty \gamma^t R_t \,\middle|\, X_0 = x, A_0 = a\right]$. For a discounted MDP, we define the *optimal value* and *action-value* functions by $V^*(x) = \sup_\pi V^\pi(x)$ $(\forall x \in \mathcal{X})$ and $Q^*(x, a) = \sup_\pi Q^\pi(x, a)$ $(\forall x \in \mathcal{X}, \forall a \in \mathcal{A})$. We say that a policy $\pi^*$ is *optimal* if it achieves the best values in every state, i.e., if $V^{\pi^*} = V^*$. We say that a policy $\pi$ is *greedy* w.r.t. an action-value function $Q$ and write $\pi = \hat{\pi}(\cdot; Q)$, if $\pi(x) \in \arg\max_{a \in \mathcal{A}} Q(x, a)$ holds for all $x \in \mathcal{X}$. Similarly, the policy $\pi$ is greedy w.r.t. $V$, if for all $x \in \mathcal{X}$, $\pi(x) \in \mathrm{argmax}_{a \in \mathcal{A}} \int P(dx'|x, a)[r(x, a) + \gamma V(x')]$ (If there exist multiple maximizers, some maximizer is chosen in an arbitrary deterministic manner). Greedy policies are important because a greedy policy w.r.t. $Q^*$ (or $V^*$) is an optimal policy. Hence, knowing $Q^*$ is sufficient for behaving optimally (cf. Proposition 4.3 of [19]).

We define the Bellman operator for a policy $\pi$ as $(T^\pi V)(x) \triangleq r(x, \pi(x)) + \gamma \int V^\pi(x') P(dx'|x, a)$ and $(T^\pi Q)(x, a) \triangleq r(x, a) + \gamma \int Q(x', \pi(x')) P(dx'|x, a)$. Similarly, the Bellman optimality operator is defined as $(T^* V)(x) \triangleq \max_a \left\{ r(x, a) + \gamma \int V(x') P(dx'|x, a) \right\}$ and $(T^* Q)(x, a) \triangleq r(x, a) + \gamma \int \max_{a'} Q(x', a') P(dx'|x, a)$.

For a measurable space $\mathcal{X}$, with a $\sigma$-algebra $\sigma_\mathcal{X}$, we define $\mathcal{M}(\mathcal{X})$ as the set of all probability measures over $\sigma_\mathcal{X}$. For a probability measure $\rho \in \mathcal{M}(\mathcal{X})$ and the transition kernel $P^\pi$, we define $\rho P^\pi(dx') = \int P(dx'|x, \pi(x)) d\rho(x)$. In words, $\rho(P^\pi)^m \in \mathcal{M}(\mathcal{X})$ is an $m$-step-ahead probability distribution of states if the starting state distribution is $\rho$ and we follow $P^\pi$ for $m$ steps. In what follows we shall use $\|V\|_{p,\nu}$ to denote the $L^p(\nu)$-norm of a measurable function $V : \mathcal{X} \to \mathbb{R}$: $\|V\|_{p,\nu}^p \triangleq \nu |V|^p \triangleq \int_\mathcal{X} |V(x)|^p d\nu(x)$. For a function $Q : \mathcal{X} \times \mathcal{A} \mapsto \mathbb{R}$, we define $\|Q\|_{p,\nu}^p \triangleq \frac{1}{|\mathcal{A}|} \sum_{a \in \mathcal{A}} \int_\mathcal{X} |Q(x, a)|^p d\nu(x)$.

## 3   Approximate Policy Iteration

Consider the API procedure and the sequence $Q_0 \to \pi_1 \to Q_1 \to \pi_2 \to \cdots \to Q_{K-1} \to \pi_K$, where $\pi_k$ is the greedy policy w.r.t. $Q_{k-1}$ and $Q_k$ is the approximate action-value function for policy $\pi_k$. For the sequence $\{Q_k\}_{k=0}^{K-1}$, denote the **B**ellman **R**esidual (BR) and policy **A**pproximation **E**rror (AE) at each iteration by

$$\varepsilon_k^{\text{BR}} = Q_k - T^{\pi_k} Q_k, \tag{1}$$

$$\varepsilon_k^{\text{AE}} = Q_k - Q^{\pi_k}. \tag{2}$$

The goal of this section is to study the effect of $\nu$-weighted $L_{2p}$ norm of the Bellman residual sequence $\{\varepsilon_k^{\text{BR}}\}_{k=0}^{K-1}$ or the policy evaluation approximation error sequence $\{\varepsilon_k^{\text{AE}}\}_{k=0}^{K-1}$ on the performance loss $\|Q^* - Q^{\pi_K}\|_{p,\rho}$ of the outcome policy $\pi_K$.

The choice of $\rho$ and $\nu$ is arbitrary, however, a natural choice for $\nu$ is the sampling distribution of the data, which is used by the policy evaluation module. On the other hand, the probability distribution $\rho$ reflects the importance of various regions of the state space and is selected by the practitioner. One common choice, though not necessarily the best, is the stationary distribution of the optimal policy.

Because of the dynamical nature of MDP, the performance loss $\|Q^* - Q^{\pi_K}\|_{p,\rho}$ depends on the difference between the sampling distribution $\nu$ and the future-state distribution in the form of $\rho P^{\pi_1} P^{\pi_2} \cdots$. The precise form of this dependence will be formalized in Theorems 3 and 4. Before stating the results, we require to define the following *concentrability* coefficients.

**Definition 2** (Expected Concentrability of the Future-State Distribution). *Given $\rho, \nu \in \mathcal{M}(\mathcal{X})$, $\nu \ll \lambda$[1] ($\lambda$ is the Lebesgue measure), $m \geq 0$, and an arbitrary sequence of stationary policies $\{\pi_m\}_{m \geq 1}$, let $\rho P^{\pi_1} P^{\pi_2} \ldots P^{\pi_m} \in \mathcal{M}(\mathcal{X})$ denote the future-state distribution obtained when the first state is distributed according to $\rho$ and then we follow the sequence of policies $\{\pi_k\}_{k=1}^m$.*

*Define the following concentrability coefficients that is used in API analysis:*

$$c_{PI_1,\rho,\nu}(m_1, m_2; \pi) \triangleq \left( \mathbb{E}_{X \sim \nu} \left[ \left| \frac{d\left(\rho(P^{\pi^*})^{m_1}(P^\pi)^{m_2}\right)}{d\nu}(X) \right|^2 \right] \right)^{\frac{1}{2}},$$

$$c_{PI_2,\rho,\nu}(m_1, m_2; \pi_1, \pi_2) \triangleq \left( \mathbb{E}_{X \sim \nu} \left[ \left| \frac{d\left(\rho(P^{\pi^*})^{m_1}(P^{\pi_1})^{m_2} P^{\pi_2}\right)}{d\nu}(X) \right|^2 \right] \right)^{\frac{1}{2}},$$

$$c_{PI_3,\rho,\nu} \triangleq \left( \mathbb{E}_{X \sim \nu} \left[ \left| \frac{d\left(\rho P^{\pi^*}\right)}{d\nu}(X) \right|^2 \right] \right)^{\frac{1}{2}},$$

*with the understanding that if the future-state distribution $\rho(P^{\pi^*})^{m_1}(P^\pi)^{m_2}$ (or $\rho(P^{\pi^*})^{m_1}(P^{\pi_1})^{m_2}P^{\pi_2}$ or $\rho P^{\pi^*}$) is not absolutely continuous w.r.t. $\nu$, then we take $c_{PI_1,\rho,\nu}(m_1,m_2;\pi) = \infty$ (similar for others).*

*Also define the following concentrability coefficient that is used in AVI analysis:*

$$c_{VI,\rho,\nu}(m_1,m_2;\pi) \triangleq \left( \mathbb{E}_{X\sim\nu} \left[ \left| \frac{d\left(\rho(P^\pi)^{m_1}(P^{\pi^*})^{m_2}\right)}{d\nu}(X) \right|^2 \right] \right)^{\frac{1}{2}},$$

*with the understanding that if the future-state distribution $\rho(P^{\pi^*})^{m_1}(P^\pi)^{m_2}$ is not absolutely continuous w.r.t. $\nu$, then we take $c_{VI,\rho,\nu}(m_1,m_2;\pi) = \infty$.*

In order to compactly present our results, we define the following notation:

$$\alpha_k = \frac{(1-\gamma)\gamma^{K-k-1}}{1-\gamma^{K+1}} \qquad 0 \le k < K.$$

**Theorem 3** (Error Propagation for API). *Let $p \ge 1$ be a real number, $K$ be a positive integer, and $Q_{max} \le \frac{R_{max}}{1-\gamma}$. Then for any sequence $\{Q_k\}_{k=0}^{K-1} \subset B(\mathcal{X} \times \mathcal{A}, Q_{max})$ (space of $Q_{max}$-bounded measurable functions defined on $\mathcal{X} \times \mathcal{A}$) and the corresponding sequence $\{\varepsilon_k\}_{k=0}^{K-1}$ defined in (1) or (2), we have*

$$\|Q^* - Q^{\pi_K}\|_{p,\rho} \le \frac{2\gamma}{(1-\gamma)^2} \left[ \inf_{r\in[0,1]} C_{PI(BR/AE),\rho,\nu}^{\frac{1}{2p}}(K;r) \mathcal{E}^{\frac{1}{2p}}(\varepsilon_0,\ldots,\varepsilon_{K-1};r) + \gamma^{\frac{K}{p}-1} R_{max} \right].$$

*where $\mathcal{E}(\varepsilon_0,\ldots,\varepsilon_{K-1};r) = \sum_{k=0}^{K-1} \alpha_k^{2r} \|\varepsilon_k\|_{2p,\nu}^{2p}$.*

*(a) If $\varepsilon_k = \varepsilon^{BR}$ for all $0 \le k < K$, we have*

$$C_{PI(BR),\rho,\nu}(K;r) = (\frac{1-\gamma}{2})^2 \sup_{\pi_0',\ldots,\pi_K'} \sum_{k=0}^{K-1} \alpha_k^{2(1-r)} \left( \sum_{m\ge0} \gamma^m \Big( c_{PI_1,\rho,\nu}(K-k-1,m+1;\pi_{k+1}') + \right.$$

$$\left. c_{PI_1,\rho,\nu}(K-k,m;\pi_k') \Big) \right)^2.$$

*(b) If $\varepsilon_k = \varepsilon^{AE}$ for all $0 \le k < K$, we have*

$$C_{PI(AE),\rho,\nu}(K;r,s) = (\frac{1-\gamma}{2})^2 \sup_{\pi_0',\ldots,\pi_K'} \sum_{k=0}^{K-1} \alpha_k^{2(1-r)} \left( \sum_{m\ge0} \gamma^m c_{PI_1,\rho,\nu}(K-k-1,m+1;\pi_{k+1}') + \right.$$

$$\left. \sum_{m\ge1} \gamma^m c_{PI_2,\rho,\nu}(K-k-1,m;\pi_{k+1}',\pi_k') + c_{PI_3,\rho,\nu} \right)^2.$$

## 4 Approximate Value Iteration

Consider the AVI procedure and the sequence $V_0 \to V_1 \to \cdots \to V_{K-1}$, in which $V_{k+1}$ is the result of approximately applying the Bellman optimality operator on the previous estimate $V_k$, i.e., $V_{k+1} \approx T^* V_k$. Denote the approximation error caused at each iteration by

$$\varepsilon_k = T^* V_k - V_{k+1}. \tag{3}$$

The goal of this section is to analyze AVI procedure and to relate the approximation error sequence $\{\varepsilon_k\}_{k=0}^{K-1}$ to the performance loss $\|V^* - V^{\pi_K}\|_{p,\rho}$ of the obtained policy $\pi_K$, which is the greedy policy w.r.t. $V_{K-1}$.

**Theorem 4** (Error Propagation for AVI). *Let $p \geq 1$ be a real number, $K$ be a positive integer, and $V_{max} \leq \frac{R_{max}}{1-\gamma}$. Then for any sequence $\{V_k\}_{k=0}^{K-1} \subset B(\mathcal{X}, V_{max})$, and the corresponding sequence $\{\varepsilon_k\}_{k=0}^{K-1}$ defined in (3), we have*

$$\|V^* - V^{\pi_K}\|_{p,\rho} \leq \frac{2\gamma}{(1-\gamma)^2} \left[ \inf_{r \in [0,1]} C_{VI,\rho,\nu}^{\frac{1}{2p}}(K;r) \mathcal{E}^{\frac{1}{2p}}(\varepsilon_0, \ldots, \varepsilon_{K-1}; r) + \frac{2}{1-\gamma} \gamma^{\frac{K}{p}} R_{max} \right],$$

*where*

$$C_{VI,\rho,\nu}(K;r) = (\frac{1-\gamma}{2})^2 \sup_{\pi'} \sum_{k=0}^{K-1} \alpha_k^{2(1-r)} \left( \sum_{m \geq 0} \gamma^m \left( c_{VI,\rho,\nu}(m, K-k; \pi') + c_{VI,\rho,\nu}(m+1, K-k-1; \pi') \right) \right)^2,$$

*and $\mathcal{E}(\varepsilon_0, \ldots, \varepsilon_{K-1}; r) = \sum_{k=0}^{K-1} \alpha_k^{2r} \|\varepsilon_k\|_{2p,\nu}^{2p}$.*

## 5 Discussion

In this section, we discuss significant improvements of Theorems 3 and 4 over previous results such as [16, 18, 17, 7].

### 5.1 $L_p$ norm instead of $L_\infty$ norm

As opposed to most error upper bounds, Theorems 3 and 4 relate $\|V^* - V^{\pi_K}\|_{p,\rho}$ to the $L_p$ norm of the approximation or Bellman errors $\|\varepsilon_k\|_{2p,\nu}$ of iterations in API/AVI. This should be contrasted with the traditional, and more conservative, results such as $\limsup_{k \to \infty} \|V^* - V^{\pi_k}\|_\infty \leq \frac{2\gamma}{(1-\gamma)^2} \limsup_{k \to \infty} \|V^{\pi_k} - V_k\|_\infty$ for API (Proposition 6.2 of Bertsekas and Tsitsiklis [16]). The use of $L_p$ norm not only is a huge improvement over conservative supremum norm, but also allows us to benefit from the vast literature on supervised learning techniques, which usually provides error upper bounds in the form of $L_p$ norms, in the context of RL/Planning problems. This is especially interesting for the case of $p = 1$ as the performance loss $\|V^* - V^{\pi_K}\|_{1,\rho}$ is the difference between the expected return of the optimal policy and the resulted policy $\pi_K$ when the initial state distribution is $\rho$. Convenient enough, the errors appearing in the upper bound are in the form of $\|\varepsilon_k\|_{2,\nu}$ which is very common in the supervised learning literature. This type of improvement, however, has been done in the past couple of years [18, 17, 7] - see Proposition 1 in Section 1.

### 5.2 Expected versus supremum concentrability of the future-state distribution

The concentrability coefficients (Definition 2) reflect the effect of future-state distribution on the performance loss $\|V^* - V^{\pi_K}\|_{p,\rho}$. Previously it was thought that the key contributing factor to the performance loss is the supremum of the Radon-Nikodym derivative of these two distributions. This is evident in the definition of $C_{\rho,\nu}$ in Proposition 1 where we have terms in the form of $||\frac{d(\rho(P^\pi)^m)}{d\nu}||_\infty$ instead of $\left( \mathbb{E}_{X \sim \nu} \left[ |\frac{d(\rho(P^\pi)^m)}{d\nu}(X)|^2 \right] \right)^{\frac{1}{2}}$ that we have in Definition 2.

Nevertheless, it turns out that the key contributing factor that determines the performance loss is the *expectation* of the squared Radon-Nikodym derivative instead of its supremum. Intuitively this implies that even if for some subset of $\mathcal{X}' \subset \mathcal{X}$ the ratio $\frac{d(\rho(P^\pi)^m)}{d\nu}$ is large but the probability $\nu(\mathcal{X}')$ is very small, performance loss due to it is still small. This phenomenon has not been suggested by previous results.

As an illustration of this difference, consider a Chain Walk with 1000 states with a single policy that drifts toward state 1 of the chain. We start with $\rho(x) = \frac{1}{201}$ for $x \in [400, 600]$ and zero everywhere else. Then we evaluate both $||\frac{d(\rho(P^\pi)^m)}{d\nu}||_\infty$ and $(\mathbb{E}_{X \sim \nu} \left[ |\frac{d(\rho(P^\pi)^m)}{d\nu}|^2 \right])^{\frac{1}{2}}$ for $m = 1, 2, \ldots$ when $\nu$ is the uniform distribution. The result is shown in Figure 1a. One sees that the ratio is constant in the beginning, but increases when the distribution $\rho(P^\pi)^m$ concentrates around state 1, until it reaches steady-state. The growth and the final value of the expectation-based concentrability coefficient is much smaller than that of supremum-based.

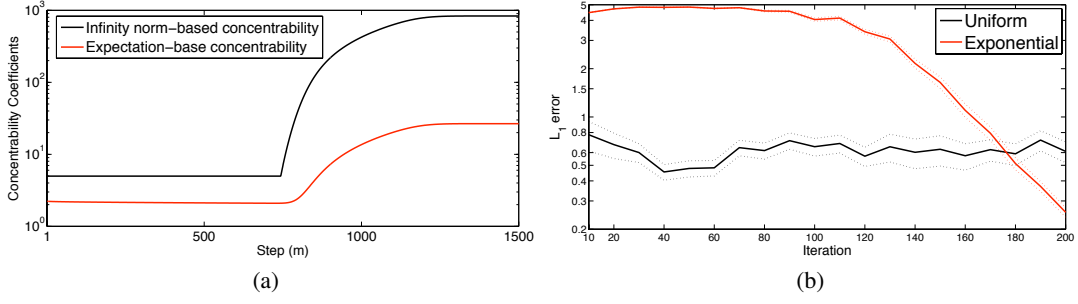

Figure 1: (a) Comparison of $\mathbb{E}_{X \sim \nu}\left[\left|\frac{d(\rho(P^\pi)^m)}{d\nu}\right|^2\right]^{\frac{1}{2}}$ and $\left\|\frac{d(\rho(P^\pi)^m)}{d\nu}\right\|_\infty$ (b) Comparison of $\|Q^* - Q_k\|_1$ for uniform and exponential data sampling schedule. The total number of samples is the same. [The $Y$-scale of both plots is logarithmic.]

It is easy to show that if the Chain Walk has $N$ states and the policy has the same concentrating behavior and $\nu$ is uniform, then $\|\frac{d(\rho(P^\pi)^m)}{d\nu}\|_\infty \to N$, while $(\mathbb{E}_{X \sim \nu}\left[\left|\frac{d(\rho(P^\pi)^m)}{d\nu}\right|^2\right])^{\frac{1}{2}} \to \sqrt{N}$ when $m \to \infty$. The ratio, therefore, would be of order $\Theta(\sqrt{N})$. This clearly shows the improvement of this new analysis in a simple problem. One may anticipate that this sharper behavior happens in many other problems too.

More generally, consider $C_\infty = \|\frac{d\mu}{d\nu}\|_\infty$ and $C_{L_2} = (\mathbb{E}_{X \sim \nu}\left[\left|\frac{d\mu}{d\nu}\right|^2\right])^{\frac{1}{2}}$. For a finite state space with $N$ states and $\nu$ is the uniform distribution, $C_\infty \leq N$ but $C_{L_2} \leq \sqrt{N}$. Neglecting all other differences between our results and the previous ones, we get a performance upper bound in the form of $\|Q^* - Q^{\pi_K}\|_{1,\rho} \leq c_1(\gamma) O(N^{1/4}) \sup_k \|\varepsilon_k\|_{2,\nu}$, while Proposition 1 implies that $\|Q^* - Q^{\pi_K}\|_{1,\rho} \leq c_2(\gamma) O(N^{1/2}) \sup_k \|\epsilon_k\|_{2,\nu}$. This difference between $O(N^{1/4})$ and $O(N^{1/2})$ shows a significant improvement.

## 5.3 Error decaying property

Theorems 3 and 4 show that the dependence of performance loss $\|V^* - V^{\pi_K}\|_{p,\rho}$ (or $\|Q^* - Q^{\pi_K}\|_{p,\rho}$) on $\{\varepsilon_k\}_{k=0}^{K-1}$ is in the form of $\mathcal{E}(\varepsilon_0, \ldots, \varepsilon_{K-1}; r) = \sum_{k=0}^{K-1} \alpha_k^{2r} \|\varepsilon_k\|_{2p,\nu}^{2p}$. This has a very special structure in that the approximation errors at later iterations have more contribution to the final performance loss. This behavior is obscure in previous results such as [17, 7] that the dependence of the final performance loss is expressed as $\mathcal{E}(\varepsilon_0, \ldots, \varepsilon_{K-1}; r) = \max_{k=0,\ldots,K-1} \|\varepsilon_k\|_{p,\nu}$ (see Proposition 1).

This property has practical and algorithmic implications too. It says that it is better to put more effort on having a lower Bellman or approximation error at later iterations of API/AVI. This, for instance, can be done by gradually increasing the number of samples throughout iterations, or to use more powerful, and possibly computationally more expensive, function approximators for the later iterations of API/AVI.

To illustrate this property, we compare two different sampling schedules on a simple MDP. The MDP is a 100-state, 2-action chain similar to Chain Walk problem in the work of Lagoudakis and Parr [5]. We use AVI with a lookup-table function representation. In the first sampling schedule, every 20 iterations we generate a fixed number of *fresh* samples by following a uniformly random walk on the chain (this means that we throw away old samples). This is the *fixed* strategy. In the *exponential* strategy, we again generate new samples every 20 iterations but the number of samples at the $k^{th}$ iteration is $ck^\gamma$. The constant $c$ is tuned such that the total number of both sampling strategy is almost the same (we give a slight margin of about $0.1\%$ of samples in favor of the fixed strategy). What we compare is $\|Q^* - Q_k\|_{1,\nu}$ when $\nu$ is the uniform distribution. The result can be seen in Figure 1b. The improvement of the exponential sampling schedule is evident. Of course, one

may think of more sophisticated sampling schedules but this simple illustration should be sufficient to attract the attention of practitioners to this phenomenon.

## 5.4 Restricted search over policy space

One interesting feature of our results is that it puts more structure and restriction on the way policies may be selected. Comparing $C_{\mathrm{PI},\rho,\nu}(K;r)$ (Theorem 3) and $C_{\mathrm{VI},\rho,\nu}(K;r)$ (Theorem 4) with $C_{\rho,\nu}$ (Proposition 1) we see that:

(1) Each concentrability coefficient in the definition of $C_{\mathrm{PI},\rho,\nu}(K;r)$ depends only on a single or two policies (e.g., $\pi'_k$ in $c_{\mathrm{PI}_1,\rho,\nu}(K-k,m;\pi'_k)$). The same is true for $C_{\mathrm{VI},\rho,\nu}(K;r)$. In contrast, the $m^{\mathrm{th}}$ term in $C_{\rho,\nu}$ has $\pi_1,\ldots,\pi_m$ as degrees of freedom, and this number is growing as $m \to \infty$.

(2) The operator $\sup$ in $C_{\mathrm{PI},\rho,\nu}$ and $C_{\mathrm{VI},\rho,\nu}$ appears *outside* the summation. Because of that, we only have $K+1$ degrees of freedom $\pi'_0,\ldots,\pi'_K$ to choose from in API and remarkably only a single degree of freedom in AVI. On the other other hand, $\sup$ appears *inside* the summation in the definition of $C_{\rho,\nu}$. One may construct an MDP that this difference in the ordering of $\sup$ leads to an arbitrarily large ratio of two different ways of defining the concentrability coefficients.

(3) In API, the definitions of concentrability coefficients $c_{\mathrm{PI}_1,\rho,\nu}$, $c_{\mathrm{PI}_2,\rho,\nu}$, and $c_{\mathrm{PI}_3,\rho,\nu}$ (Definition 2) imply that if $\rho = \rho^*$, the stationary distribution induced by an optimal policy $\pi^*$, then $c_{\mathrm{PI}_1,\rho,\nu}(m_1,m_2;\pi) = c_{\mathrm{PI}_1,\rho,\nu}(\cdot,m_2;\pi) = (\mathbb{E}_{X\sim\nu}\left[\left|\frac{d(\rho^*(P^\pi)^{m_2})}{d\nu}\right|^2\right])^{\frac{1}{2}}$ (similar for the other two coefficients). This special structure is hidden in the definition of $C_{\rho,\nu}$ in Proposition 1, and instead we have an extra $m_1$ degrees of flexibility.

**Remark 1.** *For general MDPs, the computation of concentrability coefficients in Definition 2 is difficult, as it is for similar coefficients defined in [18, 17, 7].*

## 6 Conclusion

To analyze an API/AVI algorithm and to study its statistical properties such as consistency or convergence rate, we require to (1) analyze the statistical properties of the algorithm running at each iteration, and (2) study the way the policy approximation/Bellman errors propagate and influence the quality of the resulted policy.

The analysis in the first step heavily uses tools from the Statistical Learning Theory (SLT) literature, e.g., Györfi et al. [22]. In some cases, such as AVI, the problem can be cast as a standard regression with the twist that extra care should be taken to the temporal dependency of data in RL scenario. The situation is a bit more complicated for API methods that directly aim for the fixed-point solution (such as LSTD and its variants), but still the same kind of tools from SLT can be used too – see Antos et al. [7], Maillard et al. [8].

The analysis for the second step is what this work has been about. In our Theorems 3 and 4, we have provided upper bounds that relate the errors at each iteration of API/AVI to the performance loss of the whole procedure. These bounds are qualitatively tighter than the previous results such as those reported by [18, 17, 7], and provide a better understanding of what factors contribute to the difficulty of the problem. In Section 5, we discussed the significance of these new results and the way they improve previous ones.

Finally, we should note that there are still some unaddressed issues. Perhaps the most important one is to study the behavior of concentrability coefficients $c_{\mathrm{PI}_1,\rho,\nu}(m_1,m_2;\pi)$, $c_{\mathrm{PI}_2,\rho,\nu}(m_1,m_2;\pi_1,\pi_2)$, and $c_{\mathrm{VI},\rho,\nu}(m_1,m_2;\pi)$ as a function of $m_1$, $m_2$, and of course the transition kernel $P$ of MDP. A better understanding of this question alongside a good understanding of the way each term $\varepsilon_k$ in $\mathcal{E}(\varepsilon_0,\ldots,\varepsilon_{K-1};r)$ behaves, help us gain more insight about the error convergence behavior of the RL/Planning algorithms.

## Footnotes

*Csaba Szepesvári is on leave from MTA SZTAKI. We would like to acknowledge the insightful comments by the reviewers. This work was partly supported by AICML, AITF, NSERC, and PASCAL2 under n°216886.

[1] For two measures $\nu_1$ and $\nu_2$ on the same measurable space, we say that $\nu_1$ is absolutely continuous with respect to $\nu_2$ (or $\nu_2$ dominates $\nu_1$) and denote $\nu_1 \ll \nu_2$ iff $\nu_2(A) = 0 \Rightarrow \nu_1(A) = 0$.

## References

[1] Damien Ernst, Pierre Geurts, and Louis Wehenkel. Tree-based batch mode reinforcement learning. *Journal of Machine Learning Research*, 6:503–556, 2005.

[2] Martin Riedmiller. Neural fitted Q iteration – first experiences with a data efficient neural reinforcement learning method. In *16th European Conference on Machine Learning*, pages 317–328, 2005.

[3] Amir-massoud Farahmand, Mohammad Ghavamzadeh, Csaba Szepesvári, and Shie Mannor. Regularized fitted Q-iteration for planning in continuous-space markovian decision problems. In *Proceedings of American Control Conference (ACC)*, pages 725–730, June 2009.

[4] Rémi Munos and Csaba Szepesvári. Finite-time bounds for fitted value iteration. *Journal of Machine Learning Research*, 9:815–857, 2008.

[5] Michail G. Lagoudakis and Ronald Parr. Least-squares policy iteration. *Journal of Machine Learning Research*, 4:1107–1149, 2003.

[6] Steven J. Bradtke and Andrew G. Barto. Linear least-squares algorithms for temporal difference learning. *Machine Learning*, 22:33–57, 1996.

[7] András Antos, Csaba Szepesvári, and Rémi Munos. Learning near-optimal policies with Bellman-residual minimization based fitted policy iteration and a single sample path. *Machine Learning*, 71:89–129, 2008.

[8] Odalric Maillard, Rémi Munos, Alessandro Lazaric, and Mohammad Ghavamzadeh. Finite-sample analysis of bellman residual minimization. In *Proceedings of the Second Asian Conference on Machine Learning (ACML)*, 2010.

[9] Amir-massoud Farahmand, Mohammad Ghavamzadeh, Csaba Szepesvári, and Shie Mannor. Regularized policy iteration. In D. Koller, D. Schuurmans, Y. Bengio, and L. Bottou, editors, *Advances in Neural Information Processing Systems 21*, pages 441–448. MIT Press, 2009.

[10] J. Zico Kolter and Andrew Y. Ng. Regularization and feature selection in least-squares temporal difference learning. In *ICML '09: Proceedings of the 26th Annual International Conference on Machine Learning*, pages 521–528, New York, NY, USA, 2009. ACM.

[11] Xin Xu, Dewen Hu, and Xicheng Lu. Kernel-based least squares policy iteration for reinforcement learning. *IEEE Trans. on Neural Networks*, 18:973–992, 2007.

[12] Tobias Jung and Daniel Polani. Least squares SVM for least squares TD learning. In *In Proc. 17th European Conference on Artificial Intelligence*, pages 499–503, 2006.

[13] Gavin Taylor and Ronald Parr. Kernelized value function approximation for reinforcement learning. In *ICML '09: Proceedings of the 26th Annual International Conference on Machine Learning*, pages 1017–1024, New York, NY, USA, 2009. ACM.

[14] Sridhar Mahadevan and Mauro Maggioni. Proto-value functions: A Laplacian framework for learning representation and control in markov decision processes. *Journal of Machine Learning Research*, 8:2169–2231, 2007.

[15] Alborz Geramifard, Michael Bowling, Michael Zinkevich, and Richard S. Sutton. iLSTD: Eligibility traces and convergence analysis. In B. Schölkopf, J. Platt, and T. Hoffman, editors, *Advances in Neural Information Processing Systems 19*, pages 441–448. MIT Press, Cambridge, MA, 2007.

[16] Dimitri P. Bertsekas and John N. Tsitsiklis. *Neuro-Dynamic Programming (Optimization and Neural Computation Series, 3)*. Athena Scientific, 1996.

[17] Rémi Munos. Performance bounds in $l_p$ norm for approximate value iteration. *SIAM Journal on Control and Optimization*, 2007.

[18] Rémi Munos. Error bounds for approximate policy iteration. In *ICML 2003: Proceedings of the 20th Annual International Conference on Machine Learning*, 2003.

[19] Dimitri P. Bertsekas and Steven E. Shreve. *Stochastic Optimal Control: The Discrete-Time Case*. Academic Press, 1978.

[20] Richard S. Sutton and Andrew G. Barto. *Reinforcement Learning: An Introduction (Adaptive Computation and Machine Learning)*. The MIT Press, 1998.

[21] Csaba Szepesvári. *Algorithms for Reinforcement Learning*. Morgan Claypool Publishers, 2010.

[22] László Györfi, Michael Kohler, Adam Krzyżak, and Harro Walk. *A Distribution-Free Theory of Nonparametric Regression*. Springer Verlag, New York, 2002.

